# Active Classification based on Value of Classifier

**Tianshi Gao**
Department of Electrical Engineering
Stanford University
Stanford, CA 94305
tianshig@stanford.edu

**Daphne Koller**
Department of Computer Science
Stanford University
Stanford, CA 94305
koller@cs.stanford.edu

## Abstract

Modern classification tasks usually involve many class labels and can be informed by a broad range of features. Many of these tasks are tackled by constructing a set of classifiers, which are then applied at test time and then pieced together in a fixed procedure determined in advance or at training time. We present an active classification process at the test time, where each classifier in a large ensemble is viewed as a potential observation that might inform our classification process. Observations are then selected dynamically based on previous observations, using a value-theoretic computation that balances an estimate of the expected classification gain from each observation as well as its computational cost. The expected classification gain is computed using a probabilistic model that uses the outcome from previous observations. This active classification process is applied at test time for each individual test instance, resulting in an efficient instance-specific decision path. We demonstrate the benefit of the active scheme on various real-world datasets, and show that it can achieve comparable or even higher classification accuracy at a fraction of the computational costs of traditional methods.

## 1 Introduction

As the scope of machine learning applications has increased, the complexity of the classification tasks that are commonly tackled has grown dramatically. On one dimension, many classification problems involve hundreds or even thousands of possible classes [8]. On another dimension, researchers have spent considerable effort developing new feature sets for particular applications, or new types of kernels. For example, in an image labeling task, we have the option of using GIST feature [26], SIFT feature [23], spatial HOG feature [33], Object Bank [21] and more. The benefits of combining information from different types of features can be very significant [12, 33].

To solve a complex classification problem, many researchers have resorted to ensemble methods, in which multiple classifiers are combined to achieve an accurate classification decision. For example, the Viola-Jones classifier [32] uses a cascade of classifiers, each of which focuses on different spatial and appearance patterns. Boosting [10] constructs a committee of weak classifiers, each of which focuses on different input distributions. Multiclass classification problems are very often reduced to a set of simpler (often binary) decisions, including one-vs-one [11], one-vs-all, error-correcting output codes [9, 1], or tree-based approaches [27, 13, 3]. Intuitively, different classifiers provide different "expertise" in making certain distinctions that can inform the classification task. However, as we discuss in Section 2, most of these methods use a fixed procedure determined at training time to apply the classifiers without adapting to each individual test instance.

In this paper, we take an active and adaptive approach to combine multiple classifiers/features at test time, based on the idea of value of information [16, 17, 24, 22]. At training time, we construct a rich family of classifiers, which may vary in the features that they use or the set of distinctions that they make (i.e., the subset of classes that they try to distinguish). Each of these classifiers is trained on all of the relevant training data. At test time, we dynamically select an instance-specific

subset of classifiers. We view each our pre-trained classifier as a possible observation we can make about an instance; each one adds a potential value towards our ability to classify the instance, but also has a cost. Starting from an empty set of observations, at each stage, we use a myopic value-of-information computation to select the next classifier to apply to the instance in a way that attempts to increase the accuracy of our classification state (e.g., decrease the uncertainty about the class label) at a low computational cost. This process stops when one of the suitable criteria is met (e.g., if we are sufficiently confident about the prediction). We provide an efficient probabilistic method for estimating the uncertainty of the class variable and about the expected gain from each classifier. We show that this approach provides a natural trajectory, in which simple, cheap classifiers are applied initially, and used to provide guidance on which of our more expensive classifiers is likely to be more informative. In particular, we show that we can get comparable (or even better) performance to a method that uses a large range of expensive classifiers, at a fraction of the computational cost.

## 2   Related Work

Our classification model is based on multiple classifiers, so it resembles ensemble methods like boosting [10], random forests [4] and output-coding based multiclass classification [9, 1, 29, 14]. However, these methods use a static decision process, where all classifiers have to be evaluated before any decision can be made. Moreover, they often consider a homogeneous set of classifiers, but we consider a variety of heterogeneous classifiers with different features and function forms.

Some existing methods can make classification decisions based on partial observations. One example is a cascade of classifiers [32, 28], where an instance goes through a chain of classifiers and the decision can be made at any point if the classifier response passes some threshold. Another type of method focuses on designing the stopping criteria. Schwing et al. [30] proposed a stopping criterion for random forests such that decisions can be made based on a subset of the trees. However, these methods have a fixed evaluation sequence for any instance, so there is no adaptive selection of which classifiers to use based on what we have already observed.

Instance-specific decision paths based on previous observations can be found in decision tree style models, e.g., DAGSVM [27] and tree-based methods [15, 13, 3]. Instead of making hard decisions based on individual observations like these methods, we use a probabilistic model to fuse information from multiple observations and only make decisions when it is sufficiently confident.

When observations are associated with different features, our method also performs feature selection. Instead of selecting a fixed set of features in the learning stage [34], we actively select instance-specific features in the test stage. Furthermore, our method also considers computational properties of the observations. Our selection criterion trades off between the statistical gain and the computational cost of the classifier, resulting in a computationally efficient cheap-to-expensive evaluation process. Similar ideas are hard-coded by Vedaldi et al. [31] without adaptive decisions about when to switch to which classifier with what cost. Angelova et al. [2] performed feature selection to achieve certain accuracy under some computational budget, but the selection is at training time without adaptation to individual test instances. Chai et al. [5] considered test-time feature value acquisition with a strong assumption that observations are conditionally independent given the class variable.

Finally, our work is inspired by decision-making under uncertainty based on value of information [16, 17, 24, 22]. For classification, Krause and Guestrin [19] used it to compute a conditional plan for asking the expert, trying to optimize classification accuracy while requiring as little expert interaction as possible. In machine learning, Cohn et al. [7] used active learning to select training instances to reduce the labeling cost and speedup the learning, while our work focuses on inference.

## 3   Model

We denote the instance and label pair as $(X, Y)$. Furthermore, we assume that we have been provided a set of trained classifiers $\mathcal{H}$, where $h_i \in \mathcal{H} : \mathcal{X} \to \mathbb{R}$ can be any real-valued classifiers (functions) from existing methods. For example, for multiclass classification, $h_i$ can be one-vs-all classifiers, one-vs-one classifiers and weak learners from the boosting algorithms. Note that $h_i$'s do not have to be homogeneous meaning that they can have different function forms, e.g., linear or nonlinear, and more importantly they can be trained on different types of features with various computational costs. Given an instance $x$, our goal is to infer $Y$ by sequentially selecting one $h_i$ to evaluate at a time, based on what has already been observed, until we are sufficiently confident about

$Y$ or some other stopping criterion is met, e.g., the computational constraint. The key in this process is how valuable we think a classifier $h_i$ is, so we introduce the value of a classifier as follows.

**Value of Classifier.** Let $\mathcal{O}$ be the set of classifiers that have already been evaluated (empty at the beginning). Denote the random variable $M_i = h_i(X)$ as the response/margin of the $i$-th classifier in $\mathcal{H}$ and denote the random vector for the observed classifiers as $\mathbf{M}_\mathcal{O} = [M_{o_1}, M_{o_2}, \ldots, M_{o_{|\mathcal{O}|}}]^T$, where $\forall o_i \in \mathcal{O}$. Given the actual observed values $\mathbf{m}_\mathcal{O}$ of $\mathbf{M}_\mathcal{O}$, we have a posterior $P(Y|\mathbf{m}_\mathcal{O})$ over $Y$. For now, suppose we are given a reward $R : P \to \mathbb{R}$ which takes in a distribution $P$ and returns a real value indicating how preferable $P$ is. Furthermore, we use $C(h_i|\mathcal{O})$ to denote the computational cost of evaluating classifier $h_i$ conditioned on the set of evaluated classifiers $\mathcal{O}$. This is because if $h_i$ shares the same feature with some $o_i \in \mathcal{O}$, we do not need to compute the feature again. With some chosen reward $R$ and a computational model $C(h_i|\mathcal{O})$, we define the value of an unobserved classifier as follows.

**Definition 1** *The value of classifier $V(h_i|\mathbf{m}_\mathcal{O})$ for a classifier $h_i$ given the observed classifier responses $\mathbf{m}_\mathcal{O}$ is the combination of the expected reward of the state informed by $h_i$ and the computational cost of $h_i$. Formally,*

$$
\begin{aligned}
V(h_i|\mathbf{m}_\mathcal{O}) &\triangleq \int P(m_i|\mathbf{m}_\mathcal{O}) R(P(Y|m_i, \mathbf{m}_\mathcal{O})) dm_i - \frac{1}{\tau} C(h_i|\mathcal{O}) \\
&= E_{m_i \sim P(M_i|\mathbf{m}_\mathcal{O})} \big[ R(P(Y|m_i, \mathbf{m}_\mathcal{O})) \big] - \frac{1}{\tau} C(h_i|\mathcal{O})
\end{aligned}
\tag{1}
$$

The value of classifier has two parts corresponding to the statistical and computational properties of the classifier respectively. The first part $V_R(h_i|\mathbf{m}_\mathcal{O}) \triangleq E\big[R(P(Y|m_i, \mathbf{m}_\mathcal{O}))\big]$ is the expected reward of $P(Y|m_i, \mathbf{m}_\mathcal{O})$, where the expectation is with respect to the posterior of $M_i$ given $\mathbf{m}_\mathcal{O}$. The second part $V_C(h_i|\mathbf{m}_\mathcal{O}) \triangleq -\frac{1}{\tau} C(h_i|\mathcal{O})$ is a computational penalty incurred by evaluating the classifier $h_i$. The constant $\tau$ controls the tradeoff between the reward and the cost.

Given the definition of the value of classifier, at each step of our sequential evaluations, our goal is to pick $h_i$ with the highest value:

$$
h^* = \underset{h_i \in \mathcal{H} \setminus \mathcal{O}}{\arg\max} V(h_i|\mathbf{m}_\mathcal{O}) = \underset{h_i \in \mathcal{H} \setminus \mathcal{O}}{\arg\max} V_R(h_i|\mathbf{m}_\mathcal{O}) + V_C(h_i|\mathbf{m}_\mathcal{O})
\tag{2}
$$

We introduce the building blocks of the value of classifier, i.e., the reward, the cost and the probabilistic model in the following, and then explain how to compute it.

**Reward Definition.** We propose two ways to define the reward $R : P \to \mathbb{R}$.

*Residual Entropy.* From the information-theoretical point of view, we want to reduce the uncertainty of the class variable $Y$ by observing classifier responses. Therefore, a natural way to define the reward is to consider the negative residual entropy, that is the lower the entropy the higher the reward. Formally, given some posterior distribution $P(Y|\mathbf{m}_\mathcal{O})$, we define

$$
R(P(Y|\mathbf{m}_\mathcal{O})) = -H(Y|\mathbf{m}_\mathcal{O}) = \sum_y P(y|\mathbf{m}_\mathcal{O}) \log P(y|\mathbf{m}_\mathcal{O})
\tag{3}
$$

The value of classifier under this reward definition is closely related to information gain. Specifically,

$$
\begin{aligned}
V_R(h_i|\mathbf{m}_\mathcal{O}) &= E_{m_i \sim P(M_i|\mathbf{m}_\mathcal{O})} \big[ -H(Y|m_i, \mathbf{m}_\mathcal{O}) \big] + H(Y|\mathbf{m}_\mathcal{O}) - H(Y|\mathbf{m}_\mathcal{O}) \\
&= I(Y; M_i|\mathbf{m}_\mathcal{O}) - H(Y|\mathbf{m}_\mathcal{O})
\end{aligned}
\tag{4}
$$

Since $H(Y|\mathbf{m}_\mathcal{O})$ is a constant w.r.t. $h_i$, we have

$$
h^* = \underset{h_i \in \mathcal{H}/\mathcal{O}}{\arg\max} V_R(h_i|\mathbf{m}_\mathcal{O}) + V_C(h_i|\mathbf{m}_\mathcal{O}) = \underset{h_i \in \mathcal{H}/\mathcal{O}}{\arg\max} I(Y; M_i|\mathbf{m}_\mathcal{O}) + V_C(h_i|\mathbf{m}_\mathcal{O})
\tag{5}
$$

Therefore, at each step, we want to pick the classifier with the highest mutual information with the class variable $Y$ given the observed classifier responses $\mathbf{m}_\mathcal{O}$ with a computational constraint.

*Classification Loss.* From the classification loss point of view, we want to minimize the expected loss when choosing classifiers to evaluate. Therefore, given a loss function $\Delta(y, y')$ specifying the

penalty of classifying an instance of class $y$ to $y'$, we can define the reward as the negative of the minimum expected loss:

$$R(P(Y|\mathbf{m}_\mathcal{O})) = -\min_{y'} \sum_y P(y|\mathbf{m}_\mathcal{O})\Delta(y,y') = -\min_{y'} E_{y \sim P(Y|\mathbf{m}_\mathcal{O})}\big[\Delta(y,y')\big] \qquad (6)$$

To gain some intuition about this definition, consider a 0-1 loss function, i.e., $\Delta(y,y') = \mathbf{1}\{y \neq y'\}$, then $R(P(Y|\mathbf{m}_\mathcal{O})) = -1 + \max_{y'} P(y'|\mathbf{m}_\mathcal{O})$. To maximize $R$, we want the peak of $P(Y|\mathbf{m}_\mathcal{O})$ to be as high as possible. In our experiment, these two reward definitions give similar results.

**Classification Cost.** The cost of evaluating a classifier $h$ on an instance $x$ can be broken down into two parts. The first part is the cost of computing the feature $\phi : \mathcal{X} \to \mathbb{R}^n$ on which $h$ is built, and the second is the cost of computing the function value of $h$ given the input $\phi(x)$. If $h$ shares the same feature as some evaluated classifiers in $\mathcal{O}$, then $C(h|\mathcal{O})$ only consists of the cost of evaluating the function $h$, otherwise it will also include the cost of computing the feature input $\phi$. Note that computing $\phi$ is usually much more expensive than evaluating the function value of $h$.

**Probabilistic Model.** Given a test instance $x$, we construct an instance-specific joint distribution over $Y$ and the selected observations $\mathbf{M}_\mathcal{O}$. Our probabilistic model is a mixture model, where each component corresponds to a class $Y = y$, and we use a uniform prior $P(Y)$. Starting from an empty $\mathcal{O}$, we model $P(M_i, Y)$ as a mixture of Gaussian distributions. At each step, given the selected $\mathbf{M}_\mathcal{O}$, we model the new joint distribution $P(M_i, \mathbf{M}_\mathcal{O}, Y) = P(M_i|\mathbf{M}_\mathcal{O}, Y)P(\mathbf{M}_\mathcal{O}, Y)$ by modeling the new $P(M_i|\mathbf{M}_\mathcal{O}, Y = y)$ as a linear Gaussian, i.e., $P(M_i|\mathbf{M}_\mathcal{O}, Y = y) = \mathcal{N}(\theta_y^T \mathbf{M}_\mathcal{O}, \sigma_y^2)$. As we show in Section 5, this choice of probabilistic model works well empirically. We discuss how to learn the distribution and do inference in the next section.

## 4 Learning and Inference

**Learning** $P(M_i|\mathbf{m}_\mathcal{O}, y)$. Given the subset of the training set $\{(x^{(j)}, y^{(j)} = y)\}_{j=1}^{N_y}$ corresponding to the instances from class y, we denote $m_i^{(j)} = h_i(x^{(j)})$, then our goal is to learn $P(M_i|\mathbf{m}_\mathcal{O}, y)$ from $\{(\mathbf{m}^{(j)}, y^{(j)} = y)\}_{j=1}^{N_y}$. If $\mathcal{O} = \emptyset$, then $P(M_i|\mathbf{m}_\mathcal{O}, y)$ reduces to the marginal distribution $P(M_i|y) = \mathcal{N}(\mu_y, \sigma_y^2)$, and based on maximum likelihood estimation, we have $\mu_y = \frac{1}{N_y}\sum_j m_i^{(j)}$, and $\sigma_y^2 = \frac{1}{N_y}\sum_j (m_i^{(j)} - \mu_y)^2$. If $\mathcal{O} \neq \emptyset$, we assume that $P(M_i|\mathbf{m}_\mathcal{O}, y)$ is a linear Gaussian, i.e., $\mu_y = \theta_y^T \mathbf{m}_\mathcal{O}$. Note that we also append a constant 1 to $\mathbf{m}_\mathcal{O}$ as the bias term. Since we know $\mathbf{m}_\mathcal{O}$ at test time, we estimate $\theta_y$ and $\sigma_y^2$ by maximizing the local likelihood with a Gaussian prior on $\theta_y$. Specifically, for each training instance $j$ from class $y$, let $w_j = e^{-\frac{\|\mathbf{m}_\mathcal{O} - \mathbf{m}_\mathcal{O}^{(j)}\|^2}{\beta}}$, where $\beta$ is a bandwidth parameter, then the *regularized local log likelihood* is

$$\mathcal{L}(\theta_y, \sigma_y; \mathbf{m}_\mathcal{O}) = -\lambda \parallel \theta_y \parallel_2^2 + \sum_{j=1}^{N_y} w_j \log \mathcal{N}(m_i^{(j)}; \theta_y^T \mathbf{m}_\mathcal{O}^{(j)}, \sigma_y^2) \qquad (7)$$

where we overload the notation $\mathcal{N}(x; \mu_y, \sigma_y^2)$ to mean the value of a Gaussian PDF with mean $\mu_y$ and variance $\sigma_y^2$ evaluated at $x$. Note that maximizing (7) is equivalent to locally weighted regression [6] with $\ell_2$ regularization. Maximizing (7) results in:

$$\hat{\theta}_y = \underset{\theta_y}{\operatorname{argmin}} \lambda \parallel \theta_y \parallel_2^2 + \sum_{j=1}^{N_y} w_j \parallel m_i^{(j)} - \theta_y^T \mathbf{m}_\mathcal{O}^{(j)} \parallel_2^2 = (\bar{\mathbf{M}}_\mathcal{O}^T \mathbf{W} \bar{\mathbf{M}}_\mathcal{O} + \lambda \mathbf{I})^{-1} \bar{\mathbf{M}}_\mathcal{O}^T \mathbf{W} \bar{\mathbf{M}}_i \qquad (8)$$

where $\bar{\mathbf{M}}_\mathcal{O}$ is a matrix whose $j$-th row is $\mathbf{m}_\mathcal{O}^{(j)T}$, $\mathbf{W}$ is a diagonal matrix whose diagonal entries are $w_j$'s , $\bar{\mathbf{M}}_i$ is an column vector whose $j$-th element is $m_i^{(j)}$, and $\mathbf{I}$ is an identity matrix. It is worth noting that $(\bar{\mathbf{M}}_\mathcal{O}^T \mathbf{W} \bar{\mathbf{M}}_\mathcal{O} + \lambda \mathbf{I})^{-1}\mathbf{W}$ in (8) does not depend on $i$, so it can be computed once and shared for different classifiers $h_i$'s. Finally, the estimated $\sigma_y^2$ is

$$\hat{\sigma_y}^2 = \frac{1}{\sum_{j=1}^{N_y} w_j} \sum_{j=1}^{N_y} w_j \parallel m_i^{(j)} - \hat{\theta}_y^T \mathbf{m}_\mathcal{O}^{(j)} \parallel^2 \qquad (9)$$

**Computing** $V(f_i|\mathbf{m}_{\mathcal{O}})$. Given the learned distribution, we can easily compute the two CPDs in (1), i.e., $P(M_i|\mathbf{m}_{\mathcal{O}})$ and $P(Y|m_i, \mathbf{m}_{\mathcal{O}})$. $P(M_i|\mathbf{m}_{\mathcal{O}})$ can be obtained as $P(M_i|\mathbf{m}_{\mathcal{O}}) = \sum_y P(M_i|\mathbf{m}_{\mathcal{O}}, y)P(y|\mathbf{m}_{\mathcal{O}})$, where $P(Y|\mathbf{m}_{\mathcal{O}})$ is the posterior over $Y$ given some observation $\mathbf{m}_{\mathcal{O}}$ which is tracked over iterations. Specifically, $P(Y|m_i, \mathbf{m}_{\mathcal{O}}) \propto P(m_i, \mathbf{m}_{\mathcal{O}}|Y)P(Y) = P(m_i|\mathbf{m}_{\mathcal{O}}, Y)P(\mathbf{m}_{\mathcal{O}}|Y)P(Y)$, where all terms are available by caching previous computations. Finally, to compute $V(f_i|\mathbf{m}_{\mathcal{O}})$, the computational part $V_C(f_i|\mathbf{m}_{\mathcal{O}})$ is just a lookup in a cost table, and the expected reward part $V_R(f_i|\mathbf{m}_{\mathcal{O}})$ can be rewritten as:

$$V_R(h_i|\mathbf{m}_{\mathcal{O}}) = \sum_y P(y|\mathbf{m}_{\mathcal{O}})E_{m_i \sim P(M_i|\mathbf{m}_{\mathcal{O}}, y)}\big[R(P(Y|m_i, \mathbf{m}_{\mathcal{O}}))\big] \qquad (10)$$

Therefore, each component $E_{m_i \sim P(M_i|\mathbf{m}_{\mathcal{O}}, y)}\big[R(P(Y|m_i, \mathbf{m}_{\mathcal{O}}))\big]$ is the expectation of a function of a scalar Gaussian variable. We use Gaussian quadrature [18] [1] to approximate each component expectation, and then do the weighted average to get $V_R(h_i|\mathbf{m}_{\mathcal{O}})$.

**Dynamic Inference.** Given the building blocks introduced before, one can execute the classification process in $|\mathcal{H}|$ steps, where at each step, the values of all the remaining classifiers are computed. However, this will incur a large scheduling cost. This is due to the fact that usually $|\mathcal{H}|$ is large. For example, in multiclass classification, if we include all one-vs-one classifiers into $\mathcal{H}$, $|\mathcal{H}|$ is quadratic in the number of classes. Since we are maintaining a belief over $Y$ as observations are accumulated, we can use it to make the inference process more adaptive resulting in small scheduling cost.

*Early Stopping.* Based on the posterior $P(Y|\mathbf{m}_{\mathcal{O}})$, we can make dynamic and adaptive decision about whether to continue observing new classifiers or stop the process. We propose two stopping criteria. We stop the inference process whenever either of them is met, and use the posterior over $Y$ at that point to make classification decision. The first criterion is based on the information-theoretic point of view. Given the current posterior estimation $P(Y|m_i, \mathbf{m}_{\mathcal{O}})$ and the previous posterior estimation $P(Y|\mathbf{m}_{\mathcal{O}})$, the relative entropy (KL-divergence) between them is $D\big(P(Y|\mathbf{m}_{\mathcal{O}}) \parallel P(Y|m_i, \mathbf{m}_{\mathcal{O}})\big)$. We stop the inference procedure when this divergence is below some threshold $t$. The second criterion is based on the classification point of view. We consider the gap between the probability of the current best class and that of the runner-up. Specifically, we define the *margin* given a posterior $P(Y|\mathbf{m}_{\mathcal{O}})$ as $\delta_m(P(Y|\mathbf{m}_{\mathcal{O}})) = P(y^*|\mathbf{m}_{\mathcal{O}}) - \max_{y \neq y^*} P(Y|\mathbf{m}_{\mathcal{O}})$, where $y^* = \arg\max_y P(y|\mathbf{m}_{\mathcal{O}})$. If $\delta_m(P(Y|\mathbf{m}_{\mathcal{O}})) \geq t'$, then the inference stops.

*Dynamic Pruning of Class Space.* In many cases, a class is mainly confused with a small number of other classes (the confusion matrix is often close to sparse). This implies that after observing a few classifiers, the posterior $P(Y|\mathbf{m}_{\mathcal{O}})$ is very likely to be dominated by a few modes leaving the rest with very small probability. For those classes $y$ with very small $P(y|\mathbf{m}_{\mathcal{O}})$, their contributions to the value of classifier (10) are negligible. Therefore, when computing (10), we ignore the components whose $P(y|\mathbf{m}_{\mathcal{O}})$ is below some small threshold (equivalent to setting the contribution from this component to 0). Furthermore, when $P(y|\mathbf{m}_{\mathcal{O}})$ falls below some very small threshold for a class $y$, we will not estimate the likelihood related to $y$, i.e., $P(M_i|\mathbf{m}_{\mathcal{O}}, y)$, but use a small constant.

*Dynamic Classifier Space.* To avoid computing the values of all the remaining classifiers, we can dynamically restrict the search space of classifiers to those having high expected mutual information with $Y$ with respect to the current posterior $P(Y|\mathbf{m}_{\mathcal{O}})$. Specifically, during the training, for each classifier $h_i$ we can compute the mutual information $I(M_i; B_y)$ between its response $M_i$ and a class $y$, where $B_y$ is a binary variable indicating whether an instance is from class $y$ or not. Given our current posterior $P(Y|\mathbf{m}_{\mathcal{O}})$, we tried two ways to rank the unobserved classifiers. First, we simply select the top $L$ classifiers with the highest $I(M_i; B_{\hat{y}})$, where $\hat{y}$ is the most probable class based on current posterior. Since we can sort classifiers in the training stage, this step is constant time. Another way is that for each classifier, we can compute a weighted mutual information score, i.e., $\sum_y P(y|\mathbf{m}_{\mathcal{O}})I(M_i; B_y)$, and we restrict the classifier space to those with the top $L$ scores. Note that computing the scores is very efficient, since it is just an inner product between two vectors, where $I(Y; B_y)$'s have been computed and cached before testing. Our experiments showed that these two scores have similar performances, and we used the first method to report the results.

**Analysis of Time Complexity.** At each iteration $t$, the scheduling overhead includes selecting the top $L$ candidate observations, and for each candidate $i$, learning $P(M_i|\mathbf{m}_{\mathcal{O}}, y)$ and computing

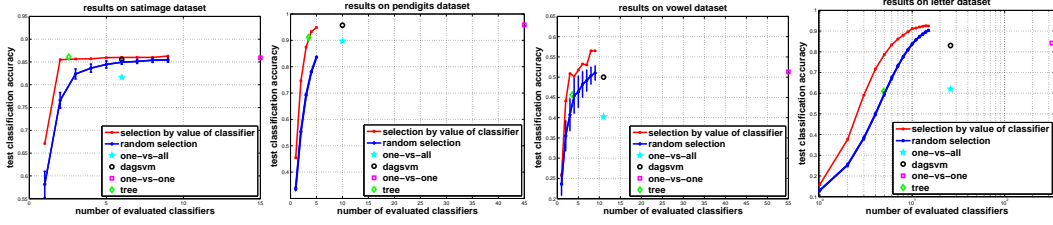

Figure 1: (Best viewed magnified and in colors) Performance comparisons on UCI datasets. From the left to right are the results on satimage, pendigits, vowel and letter (in log-scale) datasets. Note that the error bars for pendigits and letter datasets are very small (around $0.5\%$ on average).

$V(f_i|\mathbf{m}_\mathcal{O})$. First, selecting the top $L$ candidate observations is a constant time, since we can sort the observations based on $I(M_i; B_y)$ before the test process. Second, estimating $P(M_i|\mathbf{m}_\mathcal{O}, y)$ requires computing (8) and (9) for different $y$'s. Given our dynamic pruning of class space, suppose there are only $N_{t,Y}$ promising classes to consider instead of the total number of classes $K$. Since $(\bar{\mathbf{M}}_\mathcal{O}^T \mathbf{W} \bar{\mathbf{M}}_\mathcal{O} + \lambda \mathbf{I})^{-1} \mathbf{W}$ in (8) does not depend on $i$, we compute it for each promising class, which takes $O(tN_y^2 + t^2 N_y + t^3)$ floating point operations, and share it for different $i$'s. After computing this shared component, for each pair of $i$ and a promising class, computing (8) and (9) both take $O(tN_y)$. Finally, computing (10) takes $O(N_{t,Y}^2)$. Putting everything together, the overall cost at iteration $t$ is $O(N_{t,Y}(tN_y^2 + t^2 N_y + t^3) + LN_{t,Y} t N_y + LN_{t,Y}^2)$. The key to have a low cost is to effectively prune the class space (small $N_{t,Y}$) and reach a decision quickly (small $t$).

## 5 Experimental Results

We performed experiments on a collection of four UCI datasets [25] and on a scene recognition dataset [20]. All tasks are multiclass classification problems. The first set of experiments focuses on a single feature type and aims to show that (i) our probabilistic model is able to combine multiple binary classifiers to achieve comparable or higher classification accuracy than traditional methods; (ii) our active evaluation strategy successfully selects a significantly fewer number of classifiers. The second set of experiments considers multiple features, with varying computational complexities. This experiment shows the real power of our active scheme. Specifically, it dynamically selects an instance-specific subset of features, resulting in higher classification accuracy of using all features but with a significant reduction in the computational cost.

**Basic Setup.** Given a feature $\phi$, our set of classifiers $\mathcal{H}_\phi$ consists of all one-vs-one classifiers, all one-vs-all classifiers, and all node classifiers from a tree-based method [13], where a node classifier can be trained to distinguish two arbitrary clusters of classes. Therefore, for a $K$-class problem, the number of classifiers given a single feature is $|\mathcal{H}_\phi| = \frac{(K-1)K}{2} + K + N_{\phi,\text{tree}}$, where $N_{\phi,\text{tree}}$ is the number of nodes in the tree model. If there are multiple features $\{\phi_i\}_{i=1}^F$, our pool of classifiers is $\mathcal{H} = \cup_{i=1}^F \mathcal{H}_{\phi_i}$. The form of all classifiers is linear SVM for the first set of experiments and nonlinear SVM with various kernels for the second set of experiments. During training, in addition to learning the classifiers, we also need to compute the response $m_i^{(j)}$ of each classifier $h_i \in \mathcal{H}$ for each training instance $x^{(j)}$. In order to make the training distribution of the classifier responses better match the test distribution, when evaluating classifier $h_i$ on $x^{(j)}$, we do not want $h_i$ to be trained on $x^{(j)}$. To achieve this, we use a procedure similar to cross validation. Specifically, we split the training set into 10 folds, and for each fold, instances from this fold are tested using the classifiers trained on the other 9 folds. After this procedure, each training instance $x^{(j)}$ will be evaluated by all $h_i$'s. Note that the classifiers used in the test stage are trained on the entire training set. Although for different training instances $x^{(j)}$ and $x^{(k)}$ from different folds and a test instance $x$, $m_i^{(j)}$, $m_i^{(k)}$ and $m_i$ are obtained using different $h_i$'s, our experimental results confirmed that their empirical distributions are close enough to achieve good performance.

**Standard Multiclass Problems from UCI Repository.** The first set of experiments are done on four standard multiclass problems from the UCI machine learning repository [25]: vowel (speech recognition, 11 classes), letter (optical character recognition, 26 classes), satimage (pixel-based classification/segmentation on satellite images, 6 classes) and pendigits (hand written digits recognition,

10 classes). We used the same training/test split as specified in the UCI repository. For each dataset, there is only one type of feature, so it will be computed at the first step no matter which classifier is selected. After that, all classifiers have the same complexity, so the results will be independent of the $\tau$ parameter in the definition of value of classifier (1). For the baselines, we have one-vs-one with max win, one-vs-all, DAGSVM [27] and a tree-based method [13]. These methods vary both in terms of what set of classifiers they use and how those classifiers are evaluated and combined. To evaluate the effectiveness of our classifier selection scheme, we introduce another baseline that selects classifiers randomly, for which we repeated the experiments for 10 times and the average and one standard deviation are reported. We compare different methods in terms of both the classification accuracy and the number of evaluated classifiers. For our algorithm and the random selection baseline, we show the accuracy over iterations as well. Since in our framework the number of iterations (classifiers) needed varies over instances due to early stopping, the maximum number of iterations shown is defined as the mean plus one standard derivation of the number of classifier evaluations of all test instances. In addition, for the tree-based method, the number of evaluated classifiers is the mean over all test instances.

Figure 1 shows a set of results. As can be seen, our method can achieve comparable or higher accuracy than traditional methods. In fact, we achieved the best accuracy on three datasets and the gains over the runner-up methods are 0.2%, 5.2%, 8.2% for satimage, vowel, and letter datasets respectively. We think the statistical gain might come from two facts: (i) we are performing instance-specific "feature selection" to only consider those most informative classifiers; (ii) another layer of probabilistic model is used to combine the classifiers instead of the uniform voting of classifiers used by many traditional methods. In terms of the number of evaluated classifiers, our active scheme is very effective: the mean number of classifier evaluations for 6-class, 10-class, 11-class and 26-class problems are 4.50, 3.22, 6.15 and 7.72. Although the tree-based method can also use a few number of classifiers, sometimes it suffers from a significant drop in accuracy like on the vowel and letter datasets. Furthermore, compared to the random selection scheme, our method can effectively select more informative classifiers resulting in faster convergence to a certain classification accuracy.

The performance gain of our method is not free. To maintain a belief over the class variable $Y$ and to dynamically select classifiers with high value, we have introduced additional computational costs, i.e., estimating conditional distributions and computing the value of classifiers. For example, this additional cost is around 10ms for satimage, however, evaluating a linear classifier only takes less than 1ms due to very low feature dimension, so the actual running time of the active scheme is higher than one-vs-one. Therefore, our method will have a real computational advantage only if the cost of evaluating the classifiers is higher than the cost of our probabilistic inference. We demonstrate such benefit of our method in the context of multiple high dimensional features below.

**Scene Recognition.** We test our active classification on a benchmark scene recognition dataset Scene15 [20]. It has 15 scene classes and 4485 images in total. Following the protocol used in [20, 21], 100 images per class are randomly sampled for training and the remaining 2985 for test.

| model | accuracy | feature cost (# of features) | classifier cost | scheduling cost | total running time |
|---|---|---|---|---|---|
| all features | 86.40% | 52.645s (184) | 0.426s | 0 | 53.071s |
| best feature OB [21] | 83.38% | 6.20s | 0.024s | 0 | 6.224s |
| fastest feature GIST [26] | 72.70% | 0.399s | 0.0002s | 0 | 0.3992s |
| ours $\tau = 25$ | 86.26% | 1.718s (5.62) | 0.010s | 0.141s | 1.869s (**28.4x**) |
| ours $\tau = 100$ | 86.77% | 6.573s (4.71) | 0.014s | 0.116s | 6.703s (**7.9x**) |
| ours $\tau = 600$ | **88.11%** | 19.821s (4.46) | 0.031s | 0.094s | 19.946s (**2.7x**) |

Table 1: Detailed performance comparisons on Scene15 dataset with various feature types. For our methods, we show the speedup factors with respective to using all the features in a static way.

We consider various types of features, since as shown in [33], the classification accuracy can be significantly improved by combining multiple features but at a high computational cost. Our feature set includes 7 features from [33], including GIST, spatial HOG, dense SIFT, Local Binary Pattern, self-similarity, texton histogram, geometry specific histograms (please refer to [33] for details), and another recently proposed high-level image feature Object Bank [21]. The basic idea of Object Bank is to use the responses of various object detectors as the feature. The current release of the code from the authors selected 177 object detectors, each of which outputs a feature vector $\phi_i$ with

dimension 252. These individual vectors are concatenated together to form the final feature vector $\Phi = [\phi_1; \phi_2; \ldots; \phi_{177}] \in \mathbb{R}^{44,604}$. Instead of treating $\Phi$ as an undecomposable single feature vector, we can think of it as a collection of 177 different features $\{\phi_i\}_{i=1}^{177}$. Therefore, our feature pool consists of 184 features in total. Their computational costs vary from 0.035 to 13.796 seconds, with the accuracy from 54% to 83%. One traditional way to combine these features is through multiple kernel learning. Specifically, we take the average of individual kernels constructed based on individual features, and train a one-vs-all SVM using the joint average kernel. Surprisingly, this simple average kernel performs comparably with learning the weights to combine them [12].

For our active classification, we will not compute all features at the beginning of the evaluation process, but will only compute a component $\phi_i$ when a classifier $h$ based on it is selected. We will cache all evaluated $\phi_i$'s, so different classifiers sharing the same $\phi_i$ will not induce repeated computation of the common $\phi_i$. We decompose the computational costs per instance into three parts: (1) the feature cost, which is the time spent on computing the features; (2) the classifier cost, which is the time spent on evaluating the function value of the classifiers; (3) the scheduling cost, which is the time spent on selecting the classifiers using our method. To demonstrate the trade-off between the accuracy and computational cost in the definition of value of classifier, we run multiple experiments with various $\tau$'s.

The results are shown in Table 1. We also report comparisons to the best individual features in terms of either accuracy or speed (the reported accuracy is the best of one-vs-one and one-vs-all). As can be seen, combining all features using the traditional method indeed improves the accuracy significantly over those individual features, but at an expensive computational cost. However, using active classification, to achieve similar accuracy as the baseline of all features, we can get 28.4x speedup ($\tau = 25$). Note that at this configuration, our method is faster than the state-of-the-art individual feature [21], and is also 2.8% better in accuracy. Furthermore, if we put more emphasis on the accuracy, we can get the best accuracy 88.11% when $\tau = 600$.

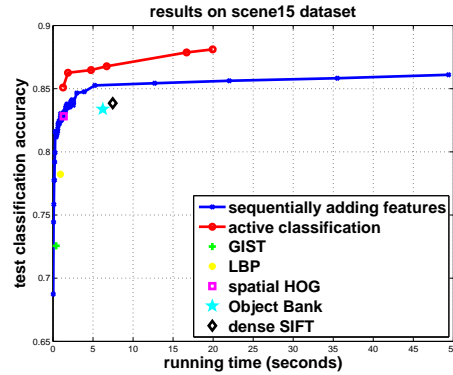

Figure 2: Classification accuracy versus running time for the baseline, active classification, and various individual features.

To further test the effectiveness of our active selection scheme, we compare with another baseline that sequentially adds one feature at a time from a filtered pool of features. Specifically, we first rank the individual features based on their classification accuracy, and only consider the top 80 features (using 80 features achieves essentially the same accuracy as using 184 features). Given this selected pool, we arrange the features in order of increasing computational complexity, and then train a classifier based on the top $N$ features for all values of $N$ from 1 to 80. As shown in Figure 2, our active scheme is one order of magnitude faster than the baseline given the same level of accuracy.

## 6 Conclusion and Future Work

In this paper, we presented an active classification process based on the value of classifier. We applied this active scheme in the context of multiclass classification, and achieved comparable and even higher classification accuracy with significant computational savings compared to traditional static methods. One interesting future direction is to estimate the value of features instead of individual classifiers. This is particularly important when computing the feature is much more expensive than evaluating the function value of classifiers, which is often the case. Once a feature has been computed, a set of classifiers that are built on it will be cheap to evaluate. Therefore, predicting the value of the feature (equivalent to the joint value of multiple classifiers sharing the same feature) can potentially lead to more computationally efficient classification process.

**Acknowledgment.** This work was supported by the NSF under grant No. RI-0917151, the Office of Naval Research MURI grant N00014-10-10933, and the Boeing company. We thank Pawan Kumar and the reviewers for helpful feedbacks.

## Footnotes

[1] We found that 3 or 5 points provide an accurate approximation.

# References

[1] E. L. Allwein, R. E. Schapire, and Y. Singer. Reducing multiclass to binary: a unifying approach for margin classifiers. *J. Mach. Learn. Res.*, 1:113–141, 2001.

[2] A. Angelova, L. Matthies, D. Helmick, and P. Perona. Fast terrain classification using variable-length representation for autonomous navigation. *CVPR*, 2007.

[3] S. Bengio, J. Weston, and D. Grangier. Label embedding trees for large multiclass task. In *NIPS*, 2010.

[4] L. Breiman. Random forests. In *Machine Learning*, pages 5–32, 2001.

[5] X. Chai, L. Deng, and Q. Yang. Test-cost sensitive naive bayes classification. In *ICDM*, 2004.

[6] W. S. Cleveland and S. J. Devlin. Locally weighted regression: An approach to regression analysis by local fitting. *Journal of the American Statistical Association*, 83:596–610, 1988.

[7] D.A. Cohn, Zoubin Ghahramani, and M.I. Jordan. Active learning with statistical models. *CoRR*, cs.AI/9603104, 1996.

[8] J. Deng, A.C. Berg, K. Li, and L. Fei-Fei. What does classifying more than 10,000 image categories tell us? In *ECCV10*, pages V: 71–84, 2010.

[9] T. G. Dietterich and G. Bakiri. Solving multiclass learning problems via error-correcting output codes. *J. of A. I. Res.*, 2:263–286, 1995.

[10] Y. Freud. Boosting a weak learning algorithm by majority. In *Computational Learning Theory*, 1995.

[11] Jerome H. Friedman. Another approach to polychotomous classification. Technical report, Department of Statistics, Stanford University, 1996.

[12] P.V. Gehler and S. Nowozin. On feature combination for multiclass object classification. In *ICCV*, 2009.

[13] G. Griffin and P. Perona. Learning and using taxonomies for fast visual categorization. In *CVPR*, 2008.

[14] V. Guruswami and A. Sahai. Multiclass learning, boosting, and error-correcting codes. In *Proc. of the Twelfth Annual Conf. on Computational Learning Theory*, 1999.

[15] T. Hastie, R. Tibshirani, and J. H. Friedman. *The elements of statistical learning: data mining, inference, and prediction.* 2009.

[16] R. A. Howard. Information value theory. *IEEE Trans. on Systems Science and Cybernetics*, 1966.

[17] R. A. Howard. Decision analysis: Practice and promise. *Management Science*, 1988.

[18] D. Koller and N. Friedman. *Probabilistic Graphical Models: Principles and Techniques.* MIT Press, 2009.

[19] A. Krause and C. Guestrin. Optimal value of information in graphical models. *Journal of Artificial Intelligence Research (JAIR)*, 35:557–591, 2009.

[20] S. Lazebnik, C. Schmid, and J. Ponce. Beyond bags of features: Spatial pyramid matching for recognizing natural scene categories. In *CVPR*, 2006.

[21] L.-J. Li, H. Su, E.P. Xing, and L. Fei-Fei. Object bank: A high-level image representation for scene classification and semantic feature sparsification. In *NIPS*, 2010.

[22] D. V. Lindley. On a Measure of the Information Provided by an Experiment. *The Annals of Mathematical Statistics*, 27(4):986–1005, 1956.

[23] D.G. Lowe. Object recognition from local scale-invariant features. In *ICCV*, 1999.

[24] V.S. Mookerjee and M.V. Mannino. Sequential decision models for expert system optimization. *IEEE Trans. on Knowledge & Data Engineering*, (5):675.

[25] D.J. Newman, S. Hettich, C.L. Blake, and C.J. Merz. Uci repository of machine learning databases, 1998.

[26] Aude Oliva and Antonio Torralba. Modeling the shape of the scene: A holistic representation of the spatial envelope. *IJCV*, 2001.

[27] J.C. Platt, N. Cristianini, and J. Shawe-taylor. Large margin dags for multiclass classification. In *NIPS*, 2000.

[28] M.J. Saberian and N. Vasconcelos. Boosting classifier cascades. In *NIPS*, 2010.

[29] Robert E. Schapire. Using output codes to boost multiclass learing problems. In *ICML*, 1997.

[30] G. A. Schwing, C. Zach, Zheng Y., and M. Pollefeys. Adaptive random forest - how many "experts" to ask before making a decision? In *CVPR*, 2011.

[31] A. Vedaldi, V. Gulshan, M. Varma, and A. Zisserman. Multiple kernels for object detection. In *ICCV*, 2009.

[32] P. Viola and M. Jones. Robust Real-time Object Detection. *IJCV*, 2002.

[33] J.X. Xiao, J. Hays, K.A. Ehinger, A. Oliva, and A.B. Torralba. Sun database: Large-scale scene recognition from abbey to zoo. In *CVPR*, 2010.

[34] Y. Yang and J. O. Pedersen. A comparative study on feature selection in text categorization. In *ICML*, pages 412–420, 1997.

